# Joint Modeling of a Matrix with Associated Text via Latent Binary Features

**XianXing Zhang**
Duke University
xianxing.zhang@duke.edu

**Lawrence Carin**
Duke University
lcarin@duke.edu

## Abstract

A new methodology is developed for joint analysis of a matrix and accompanying documents, with the documents associated with the matrix rows/columns. The documents are modeled with a focused topic model, inferring interpretable latent binary features for each document. A new matrix decomposition is developed, with latent binary features associated with the rows/columns, and with imposition of a low-rank constraint. The matrix decomposition and topic model are coupled by *sharing* the latent binary feature vectors associated with each. The model is applied to roll-call data, with the associated documents defined by the legislation. Advantages of the proposed model are demonstrated for prediction of votes on a new piece of legislation, based only on the observed text of legislation. The coupling of the text and legislation is also shown to yield insight into the properties of the matrix decomposition for roll-call data.

## 1 Introduction

The analysis of legislative roll-call data provides an interesting setting for recent developments in the joint analysis of matrices and text [23, 8]. While the roll-call data matrix is typically binary, the modeling framework is general, in that it may be readily extended to categorical, integer or real observations. The problem is made interesting because, in addition to the matrix of votes, we have access to the text of the legislation (*e.g.*, characteristic of the columns of the matrix, with each column representing a piece of legislation and each row a legislator). While roll-call data provides an interesting proving ground, the basic methodologies are applicable to any setting for which one is interested in analysis of matrices, and there is text associated with the rows or columns (*e.g.*, the text may correspond to content on a website; each column of the matrix may represent a website, and each row an individual, with the matrix representing number of visits).

The analysis of roll-call data is of significant interest to political scientists [15, 6]. In most such research the binary data are typically analyzed with a probit or logistic link function, and the underlying real matrix is assumed to have rank one. Each legislator and piece of legislation exists at a point along this one dimension, which is interpreted as characterizing a (one-dimensional) political philosophy (*e.g.*, from "conservative" to "liberal").

Roll-call data analysis have principally been interested in inferring the position of legislators in the one-dimensional latent space, with this dictated in part by the fact that the ability to perform prediction is limited. As in much matrix-completion research [17, 18], one typically can only infer votes that are missing at random. It is not possible to predict the votes of legislators on a new piece of legislation (for which, for example, an entire column of votes is missing). This has motivated the joint analysis of roll-call votes and the associated legislation [23, 8]: by modeling the latent space of the text legislation with a topic model, and making connections between topics and the latent space of the matrix decomposition, one may infer votes of an entire missing column of the matrix, assuming access to the text associated with that new legislation.

While the research in [23, 8] showed the potential of joint text-matrix analysis, there were several open questions that motivate this paper. In [23, 8] a latent Dirichlet allocation (LDA) [5] topic model was employed for the text. It has been demonstrated that LDA yields inferior perplexity scores when compared to modern Bayesian topic models, such as the focused topic model (FTM) [24]. Another significant issue with [23, 8] concerns how the topic (text) and matrix models are coupled. In [23, 8] the *frequency* with which a given topic is utilized in the text legislation is used to infer the associated matrix parameters (*e.g.*, to infer the latent feature vector associated with the respective column of the matrix). This is undesirable, because the frequency with which a topic is used in the document is characteristic of the style of writing: their may be a topic that is only mentioned briefly in the document, but that is critical to the outcome of the vote, while other topics may not impact the vote but are discussed frequently in the legislation. We also wish to move beyond the rank-one matrix assumption in [15, 6, 8].

Motivated by these limitations, in this paper the FTM is employed to model the text of legislation, with each piece of legislation characterized by a latent binary vector that defines the sparse set of associated topics. A new probabilistic low-rank matrix decomposition is developed for the votes, utilizing latent binary features; this leverages the merits of what were previously two distinct lines of matrix factorization methods [13, 17]. Unlike previous approaches, the rank is not fixed *a priori* but inferred adaptively, with theoretical justifications. For a piece of legislation, the latent binary feature vectors for the FTM and matrix decomposition are shared, yielding a new means of jointly modeling text and matrices. This linkage between text and matrices is innovative as: (*i*) it's based on whether a topic is relevant to a document/legislation, not based on the frequency with which the topic is used in the document (*i.e.*, not based on the style of writing); (*ii*) it enables interpretation of the underlying latent binary features [13, 9] based upon available text data. The rest of the paper is organized as follows. Section 2 first reviews the focused topic model, then introduces a new low-rank matrix decomposition method, and the joint model of the two. Section 3 discusses posterior inference. In Section 4 quantitative results are presented for prediction of columns of roll-call votes based on the associated text legislation, and the joint model is demonstrated qualitatively to infer meaning/insight for the characteristics of legislation and voting patterns, and Section 5 concludes.

## 2 Model and Analysis

### 2.1 Focused topic modeling

Focused topic model (FTM) [24] were developed to address a limitation of related models based on the hierarchical Dirichlet process (HDP) [21]: the HDP shares a set of "global" topics across all documents, and each topic is in general manifested with non-zero probability in each document. This property of HDP tends to yield less "focused" or descriptive topics. It is desirable to share a set of topics across all documents, but with the additional constraint that a given document only utilize a small subset of the topics; this tends to yield more descriptive/focused topics, characteristic of detailed properties of the documents. A FTM is manifested as a compound linkage of the Indian buffet process (IBP) [10] and the Dirichlet process (DP). Each document draws latent binary features from an IBP to select a finite subset of atoms/topics from the DP. In the model details, the DP is represented in terms of a normalized gamma process [7] with weighting by the binary feature vector, constituting a document-specific topic distribution in which only a subset of topics are manifested with non-zero probability.

The key components of the FTM are summarized as follows [24]:

$$b_{jt}|\pi_t \sim \text{Bernoulli}(b_{jt}|\pi_t), \quad \pi_t = \prod_{l=1}^{t} \nu_t, \quad \nu_t|\alpha_r \sim \text{Beta}(\nu_t|\alpha_r, 1)$$
$$\boldsymbol{\theta}_j|\{\boldsymbol{b}_{j:}, \boldsymbol{\lambda}\} \sim \text{Dirichlet}(\boldsymbol{\theta}_j|\boldsymbol{b}_{j:} \odot \boldsymbol{\lambda}), \quad \lambda_t|\gamma \sim \text{Gamma}(\lambda_t|\gamma, 1) \tag{1}$$

where $b_{jt}$[1] $\in \{0, 1\}$ indicates if document $j$ uses topic $t$, which is modeled as drawn from an IBP parameterized by $\alpha_r$ under the stick breaking construction [20], as shown in the first line of (1). $\boldsymbol{\lambda} = \{\lambda_t\}_{t=1}^{K_r}$ represents the relative mass on $K_r$ topics ($K_r$ could be infinite in principle); $\boldsymbol{\lambda}$ is shared across all documents, analogous to the "top layer" of the HDP. $\boldsymbol{\theta}_j$ is the topic distribution for the $j$th document, and the expression $\boldsymbol{b}_{j:} \odot \boldsymbol{\lambda}$ denotes the pointwise vector product between

$\boldsymbol{b}_{j:}$ and $\boldsymbol{\lambda}$, thereby selecting a subset of topics for document $j$ (those for which the corresponding components of $\boldsymbol{b}_{j:}$ are non-zero). The rest of the FTM is constructed similar to LDA [5], where for each token $n$ in document $j$, a topic indicator is drawn as $z_{jn}|\boldsymbol{\theta}_j \sim \text{Mult}(z_{jn}|1, \boldsymbol{\theta}_j)$. Conditional on $z_{jn}$ and the topics $\{\boldsymbol{\beta}_k\}_{k=1}^{K_r}$, a word is drawn as $w_{jn}|z_{jn}, \{\boldsymbol{\beta}_k\}_{k=1}^{K_r} \sim \text{Mult}(w_{jn}|1, \boldsymbol{\beta}_{z_{jn}})$, where $\boldsymbol{\beta}_k|\eta \sim \text{Dirichlet}(\boldsymbol{\beta}_k|\eta)$.

Although in (1) $\boldsymbol{b}_{j:}$ is mainly designed to map the global prevalence of topics across the corpus, $\boldsymbol{\lambda}$, to a within-document proportion of topic usage, $\boldsymbol{\theta}_j$, latent features $\boldsymbol{b}_{j:}$ are informative in their own right, as they indicate which subset of topics is relevant to a given document. The document-dependent topic usage $\boldsymbol{b}_{j:}$ may be more important than $\boldsymbol{\theta}_j$ when characterizing the *meaning* of a document: $\boldsymbol{\theta}_j$ specifies the frequency with which each of the selected topics is utilized in document $j$ (this is related to writing style – verbosity or parsimony – and less related to meaning); it may be more important to just know what underlying topics are used in the document, characterized by $\boldsymbol{b}_{j:}$. We therefore make the linkage between documents and an associated matrix via the $\boldsymbol{b}_{j:}$, not based on $\boldsymbol{\theta}_j$ (where [23, 8] base the document-matrix linkage via $\boldsymbol{\theta}_j$ or it's empirical estimate).

## 2.2 Matrix factorization with binary latent factors and a low-rank assumption

Binary matrix factorization (BMF) [13, 14] is a general framework in which real *latent* matrix $\mathbf{X} \in \mathbb{R}^{P \times N}$ is decomposed as $\mathbf{X} = \mathbf{L}\mathbf{H}\mathbf{R}^T$, where $\mathbf{L} \in \{0, 1\}^{P \times K_l}$, $\mathbf{R} \in \{0, 1\}^{N \times K_r}$ are binary, and $\mathbf{H} \in \mathbb{R}^{K_l \times K_r}$ is real. The rows of $\mathbf{L}$ and $\mathbf{R}$ are modeled via IBPs, parameterized by $\alpha_l$ and $\alpha_r$ respectively, and $K_l$ and $K_r$ are the truncation levels for the IBPs, which again can be infinite in principle. The *observed* matrix is $\mathbf{Y}$, which may be real, binary, or categorial [12]. The observations are modeled in an element-wise fashion: $y_{ij} = f(x_{ij})$. We focus on binary observed matrices, $\mathbf{Y} \in \{0, 1\}^{P \times N}$, and utilize $f(\cdot)$ as a probit model [2]:

$$y_{ij} = \begin{cases} 1 & \text{if } \hat{x}_{ij} \geq 0 \\ 0 & \text{if } \hat{x}_{ij} < 0 \end{cases} \tag{2}$$

with $\hat{x}_{ij} = x_{ij} + \epsilon_{ij}$, where $\epsilon_{ij} \sim \mathcal{N}(0, 1)$.

We generalize the BMF framework by imposing that $\mathbf{H}$ is low-rank. Specifically, we impose the rank-1 expansion $\mathbf{H} = \sum_{k=1}^{K_c} \boldsymbol{u}_{:k}\boldsymbol{v}_{:k}^T$, where $\boldsymbol{u}_{:k}$ and $\boldsymbol{v}_{:k}$ are column vectors (thus their outer product is a rank-1 matrix), each of them is modeled here by a Gaussian distribution:

$$\boldsymbol{u}_{:k} \sim \mathcal{N}(\boldsymbol{u}_{:k}|0, \mathbf{I}_{K_l}) \quad \boldsymbol{v}_{:k} \sim \mathcal{N}(\boldsymbol{v}_{:k}|0, \mathbf{I}_{K_r}) \tag{3}$$

and $K_c$ is the number of such rank-1 matrices such that $K_c < \min(K_l, K_r)$, *i.e.,* $\mathbf{H}$ is low-rank. To motivate this model, consider the representation $\mathbf{H} = \sum_{k=1}^{K_c} \boldsymbol{u}_{:k}\boldsymbol{v}_{:k}^T$ in the decomposition $\mathbf{X} = \mathbf{L}\mathbf{H}\mathbf{R}^T$, which implies $\mathbf{X} = \sum_{k=1}^{K_c}(\mathbf{L}\boldsymbol{u}_{:k})(\mathbf{R}\boldsymbol{v}_{:k})^T$. Therefore, we may also express $\mathbf{X} = \boldsymbol{\Psi}\boldsymbol{\Phi}^T$, with $\boldsymbol{\Psi} \in \mathbb{R}^{P \times K_c}$ and $\boldsymbol{\Phi} \in \mathbb{R}^{N \times K_c}$; the $k$th column of $\boldsymbol{\Psi}$ is defined by $\mathbf{L}\boldsymbol{u}_{:k}$ and the $k$th column of $\boldsymbol{\Phi}$ defined by $\mathbf{R}\boldsymbol{v}_{:k}$. Consequently, the low-rank assumption for $\mathbf{H}$ yields a low-rank model $\mathbf{X} = \boldsymbol{\Psi}\boldsymbol{\Phi}^T$, precisely as in [17, 18]. Thus the definition of $\boldsymbol{\Psi}$ and $\boldsymbol{\Phi}$ via the binary matrices $\mathbf{L}$ and $\mathbf{R}$ and the linkage matrix $\mathbf{H}$ merges previously two distinct lines of matrix factorization methods. In the context of the application considered here, the decomposition $\mathbf{X} = \mathbf{L}\mathbf{H}\mathbf{R}^T$ will prove convenient, as we may share the binary matrices $\mathbf{L}$ or $\mathbf{R}$ among the topic usage of available documents. The binary features in $\mathbf{L}$ and $\mathbf{R}$ are therefore characteristic of the presence/absence of underlying topics, or related latent processes, and the matrix $\mathbf{H}$ provides the mapping of how these binary features map to observed data.

However, how to specify $K_c$ remains an open question for the above low-rank construction. As a contribution of this paper, we provide a new means of imposing a low-rank model within the prior. We model the "significance" of each rank-1 term in the expansion *explicitly*, using a stochastic process $\{s_k\}_{k=1}^{K_c}$, therefore $\mathbf{H}$ can be decomposed as $\mathbf{H} = \sum_{k=1}^{K_c} s_k\boldsymbol{u}_{:k}\boldsymbol{v}_{:k}^T$, $K_c$ can be infinity in principle. As a result, the hierarchical representation in modeling the latent matrix $\mathbf{X}$ in probit model can be summarized as:

$$\hat{x}_{ij}| \left\{\boldsymbol{l}_{i:}, \boldsymbol{r}_{j:}, \{\boldsymbol{u}_{:k}, \boldsymbol{v}_{:k}, s_k\}_{k=1}^{K_c}\right\} \sim \mathcal{N}\left(\hat{x}_{ij}| \sum_{k=1}^{K_c} s_k(\boldsymbol{l}_{i:}\boldsymbol{u}_{:k})(\boldsymbol{r}_{j:}\boldsymbol{v}_{:k})^T, 1\right) \tag{4}$$

Note that $s_k$ in (4) is similar to the singular value of SVD in spirit. Intuitively, we wish to impose $|s_k|$ to decrease "fast" as the increase of index $k$, and the rank-1 matrices with large indices will have

negligible impact over (4), therefore $K_c$ plays a role similar to the truncation level in stick breaking construction for DP [11] and IBP [20]. To achieve this end, we model each $s_k$ as a Gaussian random variable with a conjugate multiplicative gamma process (MGP) placed on its precision parameter:

$$s_k|\tau_k \sim \mathcal{N}\left(s_k|0, \tau_k^{-1}\right), \quad \tau_k = \prod_{l=1}^{k} \delta_l, \quad \delta_l|\alpha_c \sim \text{Gamma}(\delta_l|\alpha_c, 1) \tag{5}$$

The MGP was originally proposed in [3] for learning sparse factor models and further extended for tree-structured sparse factor models [26] and change-point stick breaking process [25], one of its properties is that it *increasingly* shrinks $s_k$ towards zero with the increase of index $k$. Next we make the above intuition rigorous. Theorem 1 below formally states that if $s_k$ is modeled by MGP as in (5), the rank-1 expansion in (4) will converge when $K_c \rightarrow \infty$.

**Theorem 1.** *When* $\alpha_c > 1$, *the sequence* $\sum_{k=1}^{K_c} s_k(\boldsymbol{l}_{i:}\boldsymbol{u}_{:k})(\boldsymbol{r}_{j:}\boldsymbol{v}_{:k})^T$ *converges in* $\ell_2$, *as* $K_c \rightarrow \infty$.

Although in MGP $K_c$ is unbounded [3], for computational considerations we would like to truncate it to a finite value $K_c \ll \max(P, N)$, without much loss of information. As justification, the following theoretical bound is obtained, in a manner similar to its counterparts in DP [11].

**Lemma 1.** *Denoting* $M_{ij}^{K_c} = \sum_{k=K_c+1}^{\infty} s_k(\boldsymbol{l}_{i:}\boldsymbol{u}_{:k})(\boldsymbol{r}_{j:}\boldsymbol{v}_{:k})^T$, *then* $\forall \epsilon > 0$ *we have* $p\{(M_{ij}^{K_c})^2 > \epsilon\} < \frac{ab(1-1/\alpha_c)}{\epsilon \alpha_c^{K_c}}$, *where* $a = \max_k \mathbb{E}(\boldsymbol{l}_{i:}\boldsymbol{u}_{:k})^2$, $b = \max_k \mathbb{E}(\boldsymbol{r}_{j:}\boldsymbol{v}_{:k})^2$.

Lemma 1 states that, when $\alpha_c > 1$ the approximation error introduced by the truncation level $K_c$ decays exponentially fast to 0, as $K_c \rightarrow \infty$. In Section 3 an MCMC method is developed to adaptively choose $K_c$ at each iteration, which alleviates us from fixing it *a priori*. The proof of Theorem 1 and Lemma 1 can be found in the Supplemental Material.

## 2.3 Joint learning of FTM and BMF

Via the FTM and BMF framework of the previous subsections, each piece of legislation $j$ is represented as two latent binary feature vectors $\boldsymbol{b}_{j:}$ and $\boldsymbol{r}_{j:}$. To jointly model the matrix of votes with associated text of legislation, a natural choice is to impose $\boldsymbol{b}_{j:} = \boldsymbol{r}_{j:}$. As a result, the full joint model can be specified by equations (1) - (5), with $b_{jt}$ in (1) replaced by $r_{jt}$. Note that the joint model links the topics characteristic of the text, to the latent binary features characteristic of legislation in the matrix decomposition; and such linkage leverages statistical strength of the two data source across the latent variables of the joint model during posterior inference. A graphical representation of the joint model can be found in the Supplemental Material.

In the context of the model for $\mathbf{Y} = f(\mathbf{X})$, with $\mathbf{X} = \mathbf{LHR}^T$, if one were to learn $\mathbf{L}$ and $\mathbf{H}$ based upon available training data, then a new legislation $\boldsymbol{y}_{:N+1}$ could be predicted if we had access to $\boldsymbol{r}_{:N+1}$. Via the construction above, not only do we gain a predictive advantage, because the new legislation's latent binary features $\boldsymbol{r}_{:N+1}$ can be obtained from modeling its document as in (1), but also the model provides powerful interpretative insights. Specifically the topics inferred from the documents may be used to interpret the latent binary features associated with the matrix factorization. These advantages will be demonstrated through experiments on legislative roll-call data in Section 4.

## 2.4 Related work

The ideal point topic model (IPTM) was developed in [8], where the supervised Latent Dirichlet Allocation (sLDA) [4] model was used to link empirical topic-usage frequencies to the latent factors via regression. In that work the dimension of the latent factors was set to 1, *e.g.,* fixing $K_c = 1$ in our nomenclature. In [23] the authors proposed to jointly analyze the voting matrix and the associated text through a mixture model, where each legislation's latent feature factor is clustered to a mixture component in coupled with that legislation's document topic distribution $\boldsymbol{\theta}$. Note that in their case each piece of legislation can only belong to one cluster, while in our case the latent binary features for each document can be effectively treated as being grouped to multiple clusters [13] (a mixed-membership model, manifested in terms of the binary feature vectors). Similar research in linking collaborative filtering and topic models can also be found in web content recommendation [1], movie recommendation[19], and scientific paper recommendation [22]. None of these methods makes use of the binary indicators as the characterization of associated documents, but perform linking via the topic distribution $\boldsymbol{\theta}$ and the latent (real) features in different ways.

# 3 Posterior Inference

We use Gibbs sampling for posterior inference over the latent variables, and only sampling equations that are unique for this model are discussed here. The rest are similar to those in [24, 13]. In the following we use $p(\cdot|-)$ to denote the conditional posterior of one variable given on all others.

**Sampling** $\{\boldsymbol{v}_{:k}, \boldsymbol{u}_{:k}\}_{k=1:K_c}$    Based on (3) and (4) the conditional posterior of $\boldsymbol{v}_{:k}$ can be written as $p(\boldsymbol{v}_{:k}|-) \propto \prod_{j=1}^{N} \mathcal{N}(\hat{\boldsymbol{x}}_{:j}| \sum_{k=1}^{K_c} s_k(\mathbf{L}\boldsymbol{u}_{:k})(\boldsymbol{r}_{j:}\boldsymbol{v}_{:k}), 1)\mathcal{N}(\boldsymbol{v}_{:k}|0, \mathbf{I}_{K_r})$. It can be shown that $p(\boldsymbol{v}_{:k}|-) = \mathcal{N}(\boldsymbol{v}_{:k}|\boldsymbol{\mu}_{\boldsymbol{v}_{:k}}, \boldsymbol{\Sigma}_{\boldsymbol{v}_{:k}})$, with mean $\boldsymbol{\mu}_{\boldsymbol{v}_{:k}} = s_k \boldsymbol{\Sigma}_{\boldsymbol{v}_{:k}} \sum_{j=1}^{N}(\mathbf{L}\boldsymbol{u}_{:k}\boldsymbol{r}_{j:})^T \tilde{\boldsymbol{x}}_{:j}^{-k}$ and covariance matrix $\boldsymbol{\Sigma}_{\boldsymbol{v}_{:k}} = [\mathbf{I}_{K_r} + s_k^2 \sum_{j=1}^{N}(\mathbf{L}\boldsymbol{u}_{:k}\boldsymbol{r}_{j:})^T(\mathbf{L}\boldsymbol{u}_{:k}\boldsymbol{r}_{j:})]^{-1}$, where $\tilde{\boldsymbol{x}}_{:j}^{-k} = \hat{\boldsymbol{x}}_{:j} - \mathbf{L}\mathbf{U}\mathbf{V}^T \boldsymbol{r}_{j:}^T + \mathbf{L}\boldsymbol{u}_{:k}\boldsymbol{r}_{j:}\boldsymbol{v}_{:k}$. By repeating the above procedure $p(\boldsymbol{u}_{:k}|-)$ can be derived similarly.

**Sampling** $\{s_k\}_{k=1:K_c}$    Based on (4) and (5) the conditional posterior of $s_k$ can be written as $p(s_k|-) \propto \prod_{j=1}^{N} \mathcal{N}(\hat{\boldsymbol{x}}_{:j}| \sum_{k=1}^{K_c} s_k(\mathbf{L}\boldsymbol{u}_{:k})(\boldsymbol{r}_{j:}\boldsymbol{v}_{:k}), 1)\mathcal{N}(s_k|0, \tau_k^{-1})$. It can be shown that $p(s_k|-) = \mathcal{N}(s_k|\mu_{s_k}, \sigma_{s_k}^2)$, with mean $\mu_{s_k} = \sigma_{s_k}^2 \sum_{j=1}^{N}((\mathbf{L}\boldsymbol{u}_{:k})(\boldsymbol{r}_{j:}\boldsymbol{v}_{:k}))^T \tilde{\boldsymbol{x}}_{:j}^{-k}$ and variance $\sigma_{s_k}^2 = 1/(\tau_k + \sum_{j=1}^{N}((\mathbf{L}\boldsymbol{u}_{:k})(\boldsymbol{r}_{j:}\boldsymbol{v}_{:k}))^T((\mathbf{L}\boldsymbol{u}_{:k})(\boldsymbol{r}_{j:}\boldsymbol{v}_{:k})))$.

**Sampling** $\{\tau_k, \delta_k\}_{k=1:K_c}$    Based on (5), given a *fixed* truncation level $K_c$ it can be sampled directly from its posterior distribution: $p(\delta_k|-) = \text{Gamma}\left(\delta_k|\alpha_c + \frac{K_c-k+1}{2}, 1 + \frac{1}{2}\sum_{l=k}^{K_c} \nu_l^{(k)} s_l^2\right)$, where $\nu_l^{(k)} = \prod_{t=1, t\neq k}^{l} \delta_t$. $\tau_k$ can then be reconstructed from $\delta_{1:k}$ as in (5).

**Sampling** $\{r_{jt}\}_{j=1:N, t=1:K_r}$    Similar to the derivation in [24], $p(r_{jt} = 1|-) = 1$ if $N_{jt} > 0$, where $N_{jt}$ denotes the number of times document $j$ used topic $t$. When $N_{jt} = 0$, based on (1) and (4) the conditional posterior of $r_{jt}$ can be written as $p(r_{jt} = 1|-) \propto \frac{\pi_t}{\pi_t + 2^{\lambda_t}(1-\pi_t)} \exp\{-\frac{1}{2}[(\mathbf{L}\boldsymbol{h}_{t:}^T)^T(\mathbf{L}\boldsymbol{h}_{t:}^T) - 2(\mathbf{L}\boldsymbol{h}_{t:}^T)^T \tilde{\boldsymbol{x}}_{:j}^{-k}]\}$, where $\boldsymbol{h}_{t:}$ represents the $t$th row of $\mathbf{H} = \sum_{k=1}^{K_c} s_k \boldsymbol{u}_{:k}\boldsymbol{v}_{:k}^T$; and $p(r_{jt} = 0|-) \propto \frac{2^{\lambda_t}(1-\pi_t)}{\pi_t + 2^{\lambda_t}(1-\pi_t)}$. $\{l_{it}\}_{i=1:P}^{t=1:K_l}$ is sampled as described in [13].

**Adaptive sampler for MGP**    The above Gibbs sampler needs a predefined truncation level $K_c$. In [3, 26] the authors proposed an adaptive sampler, tuning $K_c$ as the sampler progresses, with convergence of the chain guaranteed [16]. Specifically, the adaptation procedure is triggered with probability $p(t) = \exp(z_0 + z_1 t)$ at the $t$th iteration, with $z_0, z_1$ chosen so that adaptation occurs frequently at the beginning of the chain but decreases exponentially fast. When the adaptation is triggered in the $t$th iteration, let $q_\kappa(t) = \{k|d_\infty(s_k \mathbf{L}\boldsymbol{u}_{:k}\boldsymbol{v}_{:k}^T \mathbf{R}^T) \leq \kappa\}$ denotes in iteration $t$ the indices of the rank-1 matrices with the maximum-valued entry less than some pre-defined threshold $\kappa$, which intuitively has a negligible contribution at the $t$th iteration, and thus are deleted and $K_c$ will decrease. On the other hand, if $q_\kappa(t)$ is empty then it suggests that more rank-1 matrices are needed, in this case we increase $K_c$ by one and draw $\boldsymbol{u}_{:K_c}, \boldsymbol{v}_{:K_c}$ from their prior distributions respectively.

# 4 Experimental Results

## 4.1 Experiment setting

We have performed joint matrix and text analysis, considering the House of Representatives (House), sessions 106 - 111 [2]; we model each session's roll-call votes separately as binary matrix $\mathbf{Y}$. Entry $y_{ij} = 1$ denotes that the $i$th legislator's response to legislation $j$ is either "Yea" or "Yes", and $y_{ij} = 0$ denotes that the corresponding response is either "Nay" or "No". The data are preprocessed in the same way as described in [8]. We recommend to set the IBP hyperparameters $\alpha_l = \alpha_r = 1$, MGP hyperparameters $\alpha_c = 3$, FTM hyperparameters $\gamma = 5$ and topic model hyperparameter $\eta = 0.01$. We also considered using a random-walk MH algorithm with non-informative gamma prior to infer those hyperparameters, as described in [24, 3], and the Markov chain manifested similar mixing performance. The truncation level $K_c$ in the MGP is not fixed, but inferred from the adaptive sampler, with threshold parameter $\kappa$ set to 0.05 (it is recommended to be set small for most applications). In the study below, for each model we run 5000 iterations of the Gibbs sampler, with the first 1000 iterations discarded as burn-in, and 400 samples are collected, taking every tenth iteration afterwards, to perform Bayesian estimate on the object of interest.

## 4.2 Predicting random missing votes

In this section we study the classical problem of estimating the values of matrix data that are missing uniformly at random (in-matrix missing votes), without the use of associated documents. We compare the model proposed in (4) to the probabilistic matrix factorization (PMF) found in [17, 18]. This is done by decomposing the latent matrix $\mathbf{X} = \mathbf{\Psi}\mathbf{\Phi}^T$, where each row of $\mathbf{\Psi}$ and $\mathbf{\Phi}^T$ are drawn from a Gaussian distribution with mean and covariance matrix modeled by a Gaussian-Wishart distribution. To study the behavior of the proposed MGP prior in (5), we $(i)$ vary the number of columns (rank) $K_c$ in $\mathbf{\Psi}$ and $\mathbf{\Phi}$ as a free parameter, and call this model PMF; and $(ii)$ incorporate MGP into the decomposition of $\mathbf{X} = \mathbf{\Psi}\mathbf{S}\mathbf{\Phi}^T$ where $\mathbf{S} \in \mathbb{R}^{K_c \times K_c}$ is a diagonal matrix with each diagonal element specified as $s_k$. The model in $(ii)$ is called PMF+MGP. Additionally, to check if the low-rank assumption detailed in Section 2.2 is effective for BMF, we also compare the performance of the BMF model originally proposed in [13], which we term BMF-Original.

We compared these models on predicting the missing values selected uniformly at random, with different percentage $(90\%, 95\%, 99\%)$ of missingness. This study has been done on House data from the 106 to 111 sessions; however, to conserve space we only summarized the experimental results on the 110th House data, in Figure 1; similar results are observed across all sessions. In Figure 1 each panel corresponds to a certain percentage of missingness; the horizontal axis is the number of columns (rank), which varies as a free parameter of PMF, while the vertical axis is the prediction accuracy. MGP is observed to be generally effective in modeling the rank across three panels, and the low-rank assumption is critical to get good performance for the BMF. When the percentage of missingness is relatively low, *e.g.,* 90% or 95%, PMF performs better than BMF, however when the percentage of missingness is high *e.g.,* 99%, the BMF (with low rank assumption) is very competitive with PMF. This is probably because of the way BMF encourages the sharing of statistical strength among all rows and columns via the matrix $\mathbf{H}$ as described in [13], which is most effective when data is scarce.

## 4.3 Predicting new bills based on text

We study the predictive power of the proposed model when the legislative roll-call votes and the associated bill documents are modeled jointly, as described in Section 2.3. We compare our proposed model with the IPTM in [8], where the authors fixed the rank $K_c = 1$ in IPTM; we term this model IPTM($K_c = 1$). In [8] the authors suggested that fixing the rank to one might be over-restrictive, thus we also propose to model the rank in the ideal point model using MGP, in a similar way to how this was done for the PMF model, and call this model IPTM. We also compare our model with that in [23], where the authors proposed to combine the factor analysis model and topic model via a compounded mixture model, with all sessions of roll-call data are modeled jointly via a Markov process. Since our main goal is to predict new bills but not modeling the matrices dynamically, in the following experiments we remove the Markov process but model each session of House data separately; we call this model FATM. In [23] the authors proposed to use a beta-Bernoulli distributed binary variable $b_k$ to model if the $k$th rank-1 matrix is used in matrix decomposition. When performing posterior inference we find that $b_k$ tends to be easily trapped in local maxima, while MGP, which models the significance of usage (but not the binary usage) of each $k$th rank-1 matrix via $s_k$, smoother estimates and better mixing were observed.

For each session the bills are partitioned into 6-folds, and we iteratively remove a fold, and train the model with the remaining folds; predictions are then performed on the bills in the removed fold. The experiment results are summarized in Figure 2. Note that since $\boldsymbol{r}_{j:}$ is modeled via the stick-breaking construction of IBP as in (1), the total number of latent binary features $K_r$ is unbounded, and we face the risk of having the latent binary features important for explaining voting $\mathbf{Y}$ and important for explaining the associated text learned separately. This may lead to the undesirable consequence that the latent features learned from text are not discriminative in predicting a new piece of legislation. To reduce such risk, in practice we could either set $\alpha_r$ such that it strongly favor fewer latent binary features, or we can truncate the stick breaking construction at a pre-defined level $K_r$. For a clearer comparison with other models, where the number of topics are fixed, we choose the second approach and let $K_r$ vary as the maximum number of possible topics.

Across all sessions IPTM consistently performs better than its counterpart when $K_c = 1$; this again demonstrates the effectiveness of MGP in modeling the rank. Although there is no significant advan-

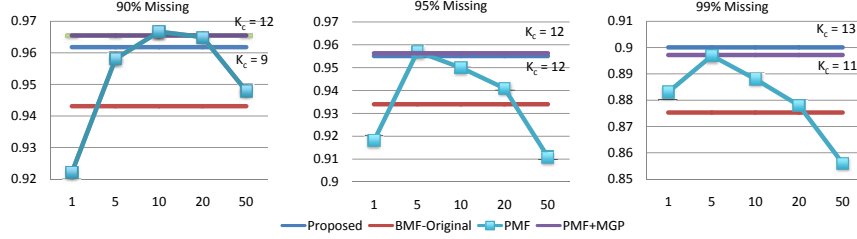

Figure 1: *Comparison of prediction accuracy for votes missing uniformly at random, for the 110th House data. Different panels corresponds to different percentage of missingness, for each panel the vertical axis represents accuracy and horizontal axis represents the rank set for PMF. For PMF+MGP and our proposed method, inferred rank $K_c$ is shown for the most-probable collection sample.*

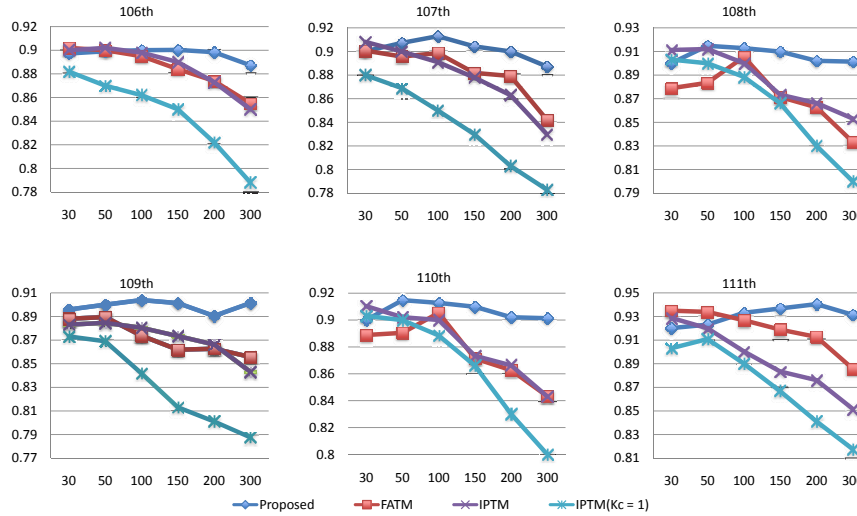

Figure 2: *Prediction accuracy for held-out legislation across 106th - 111th House data; prediction of an entire column of missing votes based on text. In each panel the vertical axis represents accuracy and the horizontal axis represents the number of topics used for each model. Results are averaged across 6-folds, with variances are too small to see.*

tage of our proposed model when the truncation on the number of topics $K_r$ (horizontal axis) is small (*e.g.*, 30-50), over-fitting is observed for all models except our proposed model. As we increase the number of topics, the performance of other models drop significantly (vertical axis). Across all five sessions, the best quantitative results are obtained by the proposed model when $K_r > 100$.

## 4.4 Latent binary feature interpretation

In this study we partition all the bills into two groups: ($i$) bills for which there is near-unanimous agreement, with "Yea" or "Yes" more than 90%; ($ii$) contentious bills with percentage of votes received as "Yea" or "Yes" less than 60%. By linking the inferred binary latent features to the topics for those two groups, we can get insight into the characteristics of legislation and voting patterns, *e.g.,* what influenced a near-unanimous yes vote, and what influenced more contention. Figure 3 compares the latent feature usage pattern of those two groups; the horizontal axis represents the latent features, where we set $K_r = 100$ for illustration purpose, and the vertical axis is the aggregated frequency that a feature/topic is used by all the bills in each of those two groups. The frequency is normalized within each group for easy interpretation. For each group, we select three discriminative features: ones heavily used in one group but rarely used in the other (these selected features are highlighted in blue/red). For example, in the left panel the features highlighted in blue are widely used by bills in the left group, but rarely used by bills in the right group. As observed

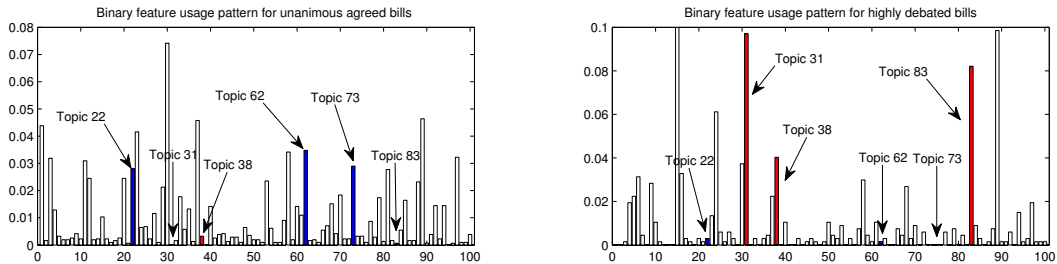

Figure 3: *Comparison of the frequencies of binary features usage between two groups of bills, left: near-unanimous affirmative bills (e.g., bills with percentage of votes received as "Yes" or "Yea" is more than 90%). Right: contentious bills (e.g., bills with percentage of votes received as "Yes" or "Yea" is less than 60%). Data from 110th House, when $K_r = 100$. The vertical axis represents the normalized frequency of using feature/topic within the corresponding group. The six most discriminative features/topics (labeled in the figure) are shown in Table 1*

Table 1: *Six discriminative topics of unanimous agreed/highly debated bills learned from the 110th house of representatives, with top-ten most probable words shown. (R) and (B) represent the topics depicted in Figure 3 as red and blue respectively.*

| TOPIC 22 (B) | TOPIC 31 (R) | TOPIC 38 (R) | TOPIC 62 (B) | TOPIC 73 (B) | TOPIC 83 (R) |
|---|---|---|---|---|---|
| CHILDREN | CONCURRENT RESOLUTION | TAX | PEOPLE | NATION | CLAUSE |
| CHILD | ADJOURN | CORPORATION | WORLD | ATTACK | PRINT |
| YOUTH | MAJORITY LEADER | TAXABLE | HOME | TERRORIST | WAIVE |
| PORNOGRAPHY | DESIGNEE | CREDIT | SANITATION | PEOPLE | SUBSTITUTE |
| INTERNET | AVIATION | PENALTY | WATER | SEPTEMBER | COMMITTEE AMENDMENT |
| FATHER | RECESS | REVENUE | INTERNATIONAL | VOLUNTEER | READ |
| FAMILY | MINORITY LEADER | TAXPAYER | SOUTHERN | CITIZEN | DEBATE |
| PARENT | FEBRUARY | SPECIAL | COMPENSATION | PAKISTAN | OFFER |
| SCHOOL | MOTION OFFER | FILE | ASSOCIATION | LEGITIMATE | DIVIDE AND CONTROL |
| EMERGENCY | STAND | SUBSTITUTE | ECONOMIC | FUTURE | MOTION |

from Figure 3, the learned binary features are discriminative, as the usage pattern for those two groups are quite different.

We also study the interpretation of those latent features by linking them to the topics inferred from the texts. As an example, those six highlighted features are linked to their corresponding topics and depicted in Table 1, with the top-ten most probable words within each topic shown. For the unanimous agreed bills, we can read from Table 1 that they are highly probable to be related to topics about the education of youth (Topic 22), or the prevention of terrorist (Topic 73). While the bills from the contentious group tend to more related to making amendments to an existing piece of legislation (Topic 83) or discussing taxation (Topic 38). Note that compared to conventional topic modeling, these inferred topics are not only informative in semantic meaning of the bills, but also discriminative in predicting the outcome of the bills.

## 5 Conclusion

A new methodology has been developed for the joint analysis of a matrix with associated text, based on sharing latent binary features modeled via the Indian buffet process. The model has been demonstrated on analysis of voting data from the US House of Representatives. Imposition of a low-rank representation for the latent real matrix has proven important, with this done in a new manner via the multiplicative gamma process. Encouraging quantitative results are demonstrated, and the model has also been shown to yield interesting insights into the meaning of the latent features. The sharing of latent binary features provides a general joint learning framework for Indian buffet process based models [9], where focused topic model and binary matrix factorization are two examples, exploring other possibilities in different scenarios could be an interesting direction.

## Acknowledgements

The authors would like to thank anonymous reviewers for providing useful comments. The research reported here was supported by ARO, DOE, NGA, ONR, and DARPA (under the MSEE program).

## Footnotes

[1]Throughout this paper notation $b_{ij}$ are used to denote the entry locates at the $i$th row and $j$th column in matrix $\mathbf{B}$, $\boldsymbol{b}_{j:}$ and $\boldsymbol{b}_{:k}$ are used to represent the $j$th row and $k$th column in $\mathbf{B}$ respectively.

[2] These data are available from thomas.loc.gov

# References

[1] D. Agarwal and B. Chen. fLDA: matrix factorization through latent Dirichlet allocation. In *WSDM*, 2010.

[2] J. H. Albert and S. Chib. Bayesian analysis of binary and polychotomous response data. *Journal of the American Statistical Association*, 1993.

[3] A. Bhattacharya and D. B. Dunson. Sparse Bayesian infinite factor models. *Biometrika*, 2011.

[4] D. M. Blei and Jon D. McAuliffe. Supervised topic models. In *Advances in Neural Information Processing Systems*, 2007.

[5] D. M. Blei, A. Ng, and M. I. Jordan. Latent Dirichlet allocation. *JMLR*, 2003.

[6] J. Clinton, S. Jackman, and D. Rivers. The statistical analysis of roll call data. *Am. Political Sc. Review*, 2004.

[7] T. Ferguson. A Bayesian analysis of some nonparametric problems. *The Annals of Statistics*, 1973.

[8] S. Gerrish and D.M. Blei. Predicting legislative roll calls from text. In *ICML*, 2011.

[9] T. L. Griffiths and Z. Ghahramani. The indian buffet process: An introduction and review. *Journal of Machine Learning Research*, 12:1185–1224, 2011.

[10] T.L. Griffiths and Z. Ghahramani. Infinite latent feature models and the Indian buffet process. In *Advances in Neural Information Processing Systems*, 2005.

[11] H. Ishwaran and L.F. James. Gibbs sampling methods for stick-breaking priors. *J. American Statistical Association*, 2001.

[12] P. McCullagh and J. Nelder. *Generalized Linear Models*. Chapman and Hall, 1989.

[13] E. Meeds, Z. Ghahramani, R. Neal, and S. Roweis. Modeling dyadic data with binary latent factors. In *Advances in Neural Information Processing Systems*. 2007.

[14] K. Miller, T. Griffiths, and M.I. Jordan. Nonparametric latent feature models for link prediction. In *Advances in Neural Information Processing Systems*, 2009.

[15] K.T. Poole. Recent developments in analytical models of voting in the U.S. congress. *Am. Political Sc. Review*, 1988.

[16] G. O. Roberts and J. S. Rosenthal. Coupling and ergodicity of adaptive MCMC. *Journal of Applied Probability*, 2007.

[17] R. Salakhutdinov and A. Mnih. Probabilistic matrix factorization. In *Advances in Neural Information Processing Systems*, 2007.

[18] R. Salakhutdinov and A. Mnih. Bayesian probabilistic matrix factorization using Markov chain Monte Carlo. In *ICML*, 2008.

[19] H. Shan and A. Banerjee. Generalized probabilistic matrix factorizations for collaborative filtering. In *ICDM*, 2010.

[20] Y. W. Teh, D. Görür, and Z. Ghahramani. Stick-breaking construction for the Indian buffet process. In *AISTATS*, 2007.

[21] Y. W. Teh, M. I. Jordan, Matthew J. Beal, and D. M. Blei. Hierarchical Dirichlet processes. *Journal of the American Statistical Association*, 2006.

[22] C. Wang and D. M. Blei. Collaborative topic modeling for recommending scientific articles. In *KDD*, 2011.

[23] E. Wang, D. Liu, J. Silva, D. B. Dunson, and L. Carin. Joint analysis of time-evolving binary matrices and associated documents. In *Advances in Neural Information Processing Systems*, 2010.

[24] S. Williamson, C. Wang, K. A. Heller, and D. M. Blei. The IBP compound Dirichlet process and its application to focused topic modeling. In *ICML*, 2010.

[25] X. Zhang, D. Dunson, and L. Carin. Hierarchical topic modeling for analysis of time-evolving personal choices. In *Advances in Neural Information Processing Systems 24*. 2011.

[26] X. Zhang, D. Dunson, and L. Carin. Tree-structured infinite sparse factor model. In *ICML*, 2011.

